# Statistical Models of Linear and Non–linear Contextual Interactions in Early Visual Processing

**Ruben Coen–Cagli**
AECOM
Bronx, NY 10461
rcagli@aecom.yu.edu

**Peter Dayan**
GCNU, UCL
17 Queen Square, LONDON
dayan@gatsby.ucl.ac.uk

**Odelia Schwartz**
AECOM
Bronx, NY 10461
oschwart@aecom.yu.edu

## Abstract

A central hypothesis about early visual processing is that it represents inputs in a coordinate system matched to the statistics of natural scenes. Simple versions of this lead to Gabor–like receptive fields and divisive gain modulation from local surrounds; these have led to influential neural and psychological models of visual processing. However, these accounts are based on an incomplete view of the visual context surrounding each point. Here, we consider an approximate model of linear and non–linear correlations between the responses of spatially distributed Gabor-like receptive fields, which, when trained on an ensemble of natural scenes, unifies a range of spatial context effects. The full model accounts for neural surround data in primary visual cortex (V1), provides a statistical foundation for perceptual phenomena associated with Li's (2002) hypothesis that V1 builds a saliency map, and fits data on the tilt illusion.

## 1  Introduction

That visual input at a given point is greatly influenced by its spatial context is manifest in a host of neural and perceptual effects (see, e.g., [1, 2]). For instance, stimuli surrounding the so-called classical receptive field (RF) lead to striking nonlinearities in the responses of visual neurons [3, 4]; spatial context results in intriguing perceptual illusions, such as the misjudgment of a center stimulus attribute in the presence of a surrounding stimulus [5–7]; it also plays a critical role in determining the salience of points in visual space, for instance controlling pop-out, contour integration, texture segmentation [8–10] and more generally locations where statistical homogeneity of the input breaks down [1]. Contextual effects are widespread across sensory systems, neural areas, and stimulus attributes — making them an attractive target for computational modeling.

There are various mechanistic treatments of extra-classical RF effects (e.g.,[11–13]) and contour integration [14], and V1's suggested role in computing salience has been realized in a large-scale dynamical model [1, 15]. There are also normative approaches to salience (e.g., [16–19]) with links to V1. However, these have not substantially encompassed neurophysiological data or indeed made connections with the perceptual literature on contour integration and the tilt illusion. Our aim is to build a principled model based on scene statistics that can ultimately account for, and therefore unify, the whole set of contextual effects above.

Much seminal work has been done in the last two decades on learning linear filters from first principles from the statistics of natural images (see e.g. [20]). However, contextual effects emerge from the interactions among multiple filters; therefore here we address the much less well studied issue of the learned, statistical, basis of the coordination of the group of filters — the scene–dependent, linear and non–linear interactions among them. We focus on recent advances in models of scene statistics, using a Gaussian Scale Mixture generative model (GSM; [21–23]) that captures the joint dependencies (e.g., [24–31]) between the activations of Gabor–like filters to natural scenes. The

GSM captures the dependencies via two components, (i) covariance in underlying Gaussian variables, which accounts for *linear* correlations in the activations of filters; and (ii) a shared mixer variable, which accounts for the *non–linear* correlations in the magnitudes of the filter activations.

As yet, the GSM has not been applied to the wide range of contextual phenomena discussed above. This is partly because linear correlations, which appear important to capture phenomena such as contour integration, have largely been ignored outside image processing (e.g., [23]). In addition, although the mixer variable of the GSM is closely related to bottom-up models of divisive normalization in cortex [32, 33], the assignment problem of grouping filters that share a common mixer for a given scene has yet to receive a computationally and neurobiologically realistic solution. Recent work has shown that incorporating a simple, predetermined, solution to the assignment problem in a GSM could capture the tilt illusion [34]. Nevertheless, the approach has not been studied in a more realistic model with Gabor-like filters, and learning assignments from natural scenes. Further, the implications of assignment for cortical V1 data and salience have not been explored.

In this paper we extend the GSM model to learn both assignments and linear covariance (section 2). We then apply the model to contextual neural V1 data, noting its link to the tilt illusion (section 3); and then to perceptual salience examples (section 4). In the discussion (section 5), we also describe the relationship between our GSM model and other recent scene statistics approaches (e.g., [31, 35]).

## 2 Methods

A recent focus in natural image statistics has been the joint conditional histograms of the activations of pairs of oriented linear filters (throughout the paper, filters come from the first level of a steerable pyramid with 4 orientations [36]). When filter pairs are proximal in space, these histograms have a characteristic *bowtie* shape: the variance of one filter depends on the magnitude of activation of the other. It has been shown [22] that this form of dependency can be captured by a class of generative model known as Gaussian Scale Mixture (GSM), which assumes that the linear filter activations $\mathbf{x} = v\mathbf{g}$ are random variables defined as the product of two other random variables, a multivariate Gaussian $\mathbf{g}$, and a (positive) scalar $v$ which scales the variance of all the Gaussian components.

Here, we address two additional properties of natural scenes. First, in addition to the variance dependency, filters which are close enough in space and feature space are linearly dependent, as shown by the tilt of the bowtie in fig. 1b. In order for the GSM to capture this effect, the multivariate Gaussian must be endowed with a non-diagonal covariance matrix. This matrix can be approximated by the sample covariance matrix of the filter activations or learned directly [23]; here, we learn it by maximizing the likelihood of the observed data. The second issue is that filter dependencies differ across image patches, implying that there is no fixed relationship between mixers and filters [28]. The general issue of learning multiple pools of filters, each assigned to a different mixer on an patch–dependent basis, has been addressed in recent work [30], but using a computationally and biologically impracticable scheme [37] which allowed for arbitrary pooling.

We consider an approximation to the assignment problem, by allowing a group of surround filters to either share or not the same mixer with a target filter. While this is clearly an oversimplified model of natural images, here we aimed for a reasonable balance between the complexity of the model, and biological plausibility of the computations involved.

### 2.1 The generative model

The basic repeating unit of our simplified model involves center and surround groups of filters: we use $n_c$ to denote the number of center filters, and $\mathbf{x_c}$ their activations; similarly, we use $n_s$ and $\mathbf{x_s}$ for the surround; finally, we define $n_{cs} \doteq n_c + n_s$ and $\mathbf{x} \doteq (x_c^1, \dots x_c^{n_c}, x_s^1, \dots x_s^{n_s})^\top$. We consider a single assignment choice as to whether the center group's mixer variable $v_c$ is (case $\xi_1$), or is not (case $\xi_2$) shared with the surround, which in the latter case would have its own mixer variable $v_s$. Thus, there are 2 configurations, or competing models, which are themselves combined (i.e., a mixture of GSM's, see also [35]). The graphical models of the two configurations are shown in Fig. 1a. We show this from the perspective of the center group, since in the implementation we will be reporting model neuron responses in the center location given the contextual surround.

Defining Gaussian components as $\mathbf{g} \doteq (g_c^1, \ldots g_c^{n_c}, g_s^1, \ldots g_s^{n_s})^\top$, and assuming the mixers are independent and the pools are independent given the mixers, the mixture distribution is:

$$p(\mathbf{x}) = p(\xi_1)p(\mathbf{x} \,|\, \xi_1) + p(\xi_2)p(\mathbf{x} \,|\, \xi_2) \tag{1}$$

$$p(\mathbf{x} \,|\, \xi_1) = \int dv_c \, p(v_c)p(\mathbf{x} \,|\, v_c, \xi_1) \tag{2}$$

$$p(\mathbf{x} \,|\, \xi_2) = \int dv_c \, p(v_c)p(\mathbf{x}_c \,|\, v_c, \xi_2) \int dv_s \, p(v_s)p(\mathbf{x}_s \,|\, v_s, \xi_2) \tag{3}$$

We assume a Rayleigh prior distribution on the mixers, and covariance matrix $\Sigma_{cs}$ for the Gaussian components for $\xi_1$, and $\Sigma_c$ and $\Sigma_s$ for center and surround, respectively, for $\xi_2$. The integrals in eqs. (2,3) can then be solved analytically:

$$p(\mathbf{x} \,|\, \xi_1) = \frac{det(\Sigma_{cs}^{-1})^{\frac{1}{2}}}{(2\pi)^{\frac{n_{cs}}{2}}} \frac{\mathcal{B}(1 - \frac{n_{cs}}{2} \,;\, \lambda_{cs})}{\lambda_{cs}^{(\frac{n_{cs}}{2}-1)}} \tag{4}$$

$$p(\mathbf{x} \,|\, \xi_2) = \frac{det(\Sigma_c^{-1})^{\frac{1}{2}}}{(2\pi)^{\frac{n_c}{2}}} \frac{\mathcal{B}(1 - \frac{n_c}{2} \,;\, \lambda_c)}{\lambda_c^{(\frac{n_c}{2}-1)}} \frac{det(\Sigma_s^{-1})^{\frac{1}{2}}}{(2\pi)^{\frac{n_s}{2}}} \frac{\mathcal{B}(1 - \frac{n_s}{2} \,;\, \lambda_s)}{\lambda_s^{(\frac{n_s}{2}-1)}} \tag{5}$$

where $\mathcal{B}$ is the modified Bessel function of the second kind, and $\lambda_{cs} = \sqrt{\mathbf{x}^\top \Sigma_{cs}^{-1} \mathbf{x}}$.

## 2.2 Learning

The parameters to be estimated are the covariance matrices ($\Sigma_{cs}$, $\Sigma_c$, $\Sigma_s$) and the prior probability ($k$) that center and surround share the same pool; we use a Generalized Expectation Maximization algorithm, specifically Multi Cycle EM [38], where a full EM cycle is divided into three subcycles, each involving a full E-step and a partial M-step performed only on one covariance matrix.

**E-step:** In the E-step we compute an estimate, $Q$, of the posterior distribution over the assignment variable, given the filter activations and the previous estimates of the parameters, namely $k^{old}$ and $\Theta^{old} \doteq \left\{ \Sigma_c, \Sigma_s, \Sigma_{cs} \right\}$. This is obtained via Bayes rule:

$$Q(\xi_1) = p(\xi_1 \,|\, \mathbf{x}, \Theta^{old}) \propto k^{old} \, p(\mathbf{x} \,|\, \xi_1, \Theta^{old}) \tag{6}$$

$$Q(\xi_2) = p(\xi_2 \,|\, \mathbf{x}, \Theta^{old}) \propto (1 - k^{old}) \, p(\mathbf{x} \,|\, \xi_2, \Theta^{old}) \tag{7}$$

**M-step:** In the M-step we increase the complete–data Log Likelihood, namely:

$$f = Q(\xi_1) \log \left[ k \, p(\mathbf{x} \,|\, \xi_1, \Theta) \right] + Q(\xi_2) \log \left[ (1 - k)p(\mathbf{x} \,|\, \xi_2, \Theta) \right] \tag{8}$$

Solving $\partial f / \partial k = 0$, we obtain $k^\star \doteq \arg\max_k [f] = Q(\xi_1)$. The other terms cannot be solved analytically, and a numerical procedure must be adopted to maximize $f$ w.r.t. the covariance matrices. This requires an explicit form for the gradient:

$$\frac{\partial f}{\partial \Sigma_{cs}^{-1}} = Q(\xi_1)\left( \frac{\Sigma_{cs}}{2} - \frac{1}{2\lambda_{cs}} \frac{\mathcal{B}(-\frac{n_{cs}}{2} \,;\, \lambda_{cs})}{\mathcal{B}(1 - \frac{n_{cs}}{2} \,;\, \lambda_{cs})} \mathbf{x}\mathbf{x}^\top \right) \tag{9}$$

Similar expressions hold for the other partial derivatives. In practice, we add the constraint that the covariances of the surround filters are spatially symmetric.

## 2.3 Inference: patch–by–patch assignment and model neural unit

Upon convergence of EM, the covariance matrices and prior $k$ over the assignment are found. Then, for a new image patch, the probability $p(\xi_1 \,|\, \mathbf{x})$ that the surround shares a common mixer with the center is inferred. The output of the center group is taken to be the estimate (for the present, we consider just the mean) of the Gaussian component $E\left[\mathbf{g}_c \,|\, \mathbf{x}\right]$, which we take to be our model neural unit response. To estimate the normalized response of the center filter, we need to compute the following expected value under the full model:

$$E\left[\mathbf{g}_c \,|\, \mathbf{x}\right] = \int d\mathbf{g}_c \, \mathbf{g}_c \, p(\mathbf{g}_c \,|\, \mathbf{x}) = p(\xi_1 \,|\, \mathbf{x})E\left[\mathbf{g}_c \,|\, \mathbf{x}, \xi_1\right] + p(\xi_2 \,|\, \mathbf{x})E\left[\mathbf{g}_c \,|\, \mathbf{x}_c, \xi_2\right] \tag{10}$$

the r.h.s., obtained by a straightforward calculation applying Bayes rule and the conditional independence of $\mathbf{x}_s$ from $\mathbf{x}_c, \mathbf{g}_c$ given $\xi_2$, is the sum of the expected value of $\mathbf{g}_c$ in the two configurations,

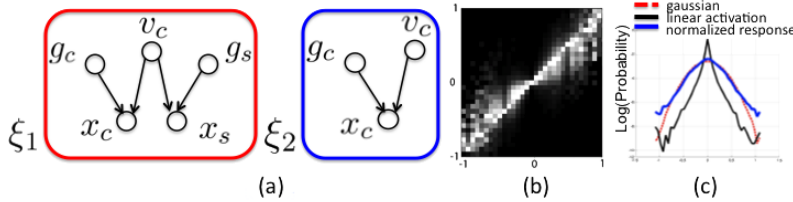

Figure 1: (a) Graphical model for the two components of the mixture of GSMs, where the center filter is ($\xi_1$; left) or is not ($\xi_2$; right) normalized by the surround filters; (b) joint conditional histogram of two linear filters activations, showing the typical *bowtie* shape due to the variance dependency, as well as a tilt due to linear dependencies between the two filters; (c) marginal distribution of linear activations in black, estimated Gaussian component in blue, and ideal Gaussian in red. The estimated distribution is closer to a Gaussian than that of the original filter.

weighted by their posterior probabilities. The explicit form for the estimate of the $i$-th component (corresponding in the implementation to a given orientation and phase) of $\mathbf{g}_c$ under $\xi_1$ is:

$$E\left[g_c^i \,|\, \mathbf{x}\,,\, \xi_1\right] = sign(x_c^i)\,\sqrt{|x_c^i|}\,\sqrt{\frac{|x_c^i|}{\lambda_{cs}}}\,\frac{\mathcal{B}(\frac{1}{2} - \frac{n_{cs}}{2}\,;\,\lambda_{cs})}{\mathcal{B}(1 - \frac{n_{cs}}{2}\,;\,\lambda_{cs})} \tag{11}$$

and a similar expression holds under $\xi_2$, replacing the subscript $cs$ by $c$. Note that in either configuration, the mixer variable's effect on this is a form of divisive normalization or gain control, through $\lambda$ (including for stability, as in [30], an additive constant set to 1 for the $\lambda$ values; we omit the formulæ to save space). Under $\xi_1$, but not $\xi_2$, this division is influenced by the surround[1]. Note also that, due to the presence of the inverse covariance matrix in $\lambda_{cs} = \sqrt{\mathbf{x}^\top \Sigma_{cs}^{-1} \mathbf{x}}$, the gain control signal is reduced when there is strong covariance, which in turn enhances the neural unit response.

## 3 Cortical neurophysiology simulations

To simulate neurophysiological experiments, we consider the following filter configuration: $3 \times 3$ spatial positions separated by 6 pixels, 2 phases (quadrature pair), and one orientation (vertical), plus 3 additional orientations in the central position to allow for cross–orientation gain control. We first learn the parameters of the model for 25000 patches from an ensemble of 5 standard scenes (Einstein, Goldhill, and so on). We take as our model neuron the absolute value of the complex activation composed by the non–linear responses (eq. (11)) of two phases of the central vertical filter.

We characterize the neuron's basic properties with a procedure that is common in physiology experiments focusing on contextual non–linear modulation. First, we measure the so-called Area Summation curve, namely the response to gratings that are optimal in orientation and spatial frequency, as a function of size. Cat and monkey experiments have shown striking non–linearities, with the peak response at low contrasts being for significantly larger diameters than at high contrasts (Figure 2a). We obtain the same behavior in the model (Figure 2b; see also [33]). This behavior is due to the assignment: for small grating sizes, center and surround have a higher posterior probability at high contrast than at low contrast, and therefore the surround exerts stronger gain control. In a reduced model with no assignment, we obtain a much weaker effect (Figure 2c).

We then assess the modulatory effect of a surround grating on a fixed, optimally–oriented central grating, as a function of their relative orientations (Figure 3a). As is common, we determine the spatial extent of the center and surround stimuli based on the area summation curves (see [4]). The model simulations (Figure 3b), as in the data, exhibit the most reduced responses when the center and surround have similar orientation (but note the "blip" when they are exactly equal in Figure 3a;b, which arises in the model from the covariance of the Gaussian; see also [31]). In addition,

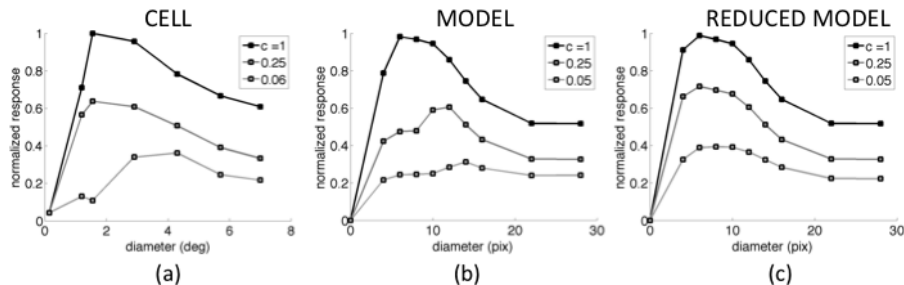

Figure 2: Area summation curves show the normalized firing rate of a neuron in response to optimal gratings of increasing size. (a) a V1 neuron, after [4]; (b) the model neuron described in Sec. 3; (c) a reduced model assuming that the surround filters are always in the gain pool of the center filter.

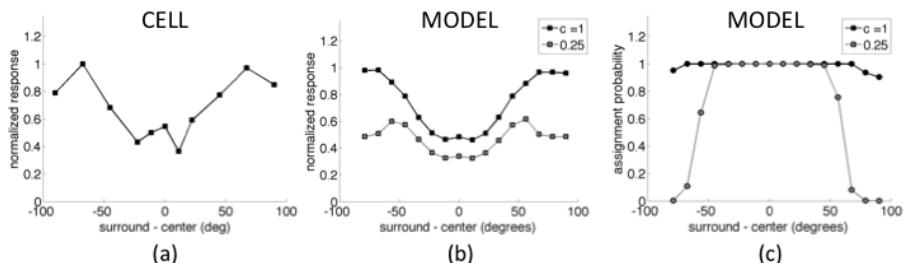

Figure 3: Orientation tuning of the surround. (a) and (b): normalized firing rate in response to a stimulus composed by an optimal central grating surrounded by an annular grating of varying orientation, for (a) a V1 neuron, after [3]; and (b) the model neuron described in Sec. 3 (c) Probability that the surround normalizes the center as a function of the relative orientation of the annular grating.

as the orientation difference between center and surround grows, the response increases and then decreases, an effect that arises from the assignments. In the model simulations, we find that the strength of this behavior depends on contrast, being larger at low contrasts, an effect of which there are hints in the neurophysiology experimental data (using contrasts of 0.2 to 0.4) but which has yet to be systematically explored. Figure 3c shows the posterior assignment probability for the same two contrasts as in figure 3b, as a function of the surround orientation. These remain close to 1 at all orientations at high contrast, but fall off more rapidly at low contrast.

Note that a previous GSM population model assumed (but did not learn) this form of fall off of the posterior weights of of figure 3c, and showed that it is a basis for explaining the so-called direct and indirect biases in the tilt illusion; i.e., repulsion and attraction in the perception of a center stimulus orientation in the presence of a surround stimulus [34]. Figure 4 compares the GSM model of [34] designed with parameters matched to perceptual data, to the result of our learned model. The qualitative shape (although not the quantitative strength) of the effects are similar.

## 4  Salience popout and contour integration simulations

To address perceptual salience effects, we need a population model of oriented units. We consider one distinct group of filters – arranged as for the model neuron in Sec. 3 – for each of four orientations ($0, 45, 90, 135$ deg, sampling more coarsely than [1]). We compute the non–linear response of each model neuron as in Sec. 3, and take the maximum across the four orientations as the population output, as in standard population decoding. This is performed at each pixel of the input image, and the result is interpreted as a *saliency* map.

We first consider the popout of a target that differs from a background of distractors by a single feature (eg. [8]), in our case orientation. Input image and output saliency map (the brighter, the more salient) are shown in Fig. 5. As in [1], the target pops out since it is less suppressed by its own, orthogonally-oriented neighbors than the surround bars are by their parallel ones; here, this emerges straight from normative inference. [8] quantified relative target saliency above detection threshold

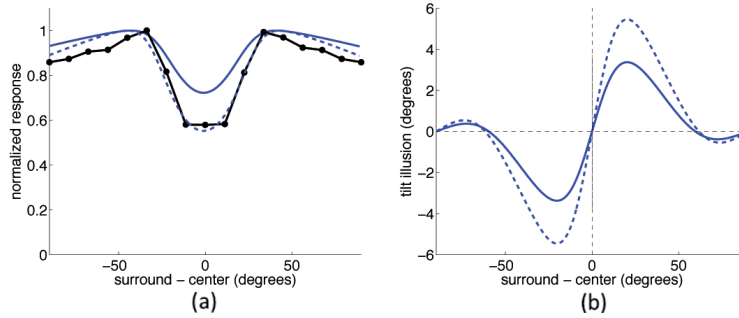

Figure 4: The tilt illusion. (a) Comparison of the learned GSM model (black, solid line with filled squares), with the GSM model in [34] (blue, solid line; parameters set to account for the illusion data of [39]), and the model in [34] with parameters modified to match the learned model (blue, dashed line). The response of each neuron in the population is plotted as a function of the difference between the surround stimulus orientation and the preferred center stimulus orientation. We assume all oriented neurons have identical properties to the learned vertical neuron (i.e., ignoring the oblique effect). The model of [34] includes idealized tuning curves. The learned model is as in the previous section, but with filters of narrower orientation tuning (because of denser sampling of 16 orientations in the pyramid), which results in an earlier point on the x axis of maximal response. Model simulations are normalized to a maximum of 1. (b) Simulations of the tilt illusion using the model in [34], based on parameters matched to the learned model (dashed line) versus parameters matched to the data of [39] (solid line).

as a function of the difference in orientation between target and distractors using luminance (Fig. 5b). Fig. 5c plots saliency from the model; it exhibits non–linear saturation for large orientation contrast, an effect that not all saliency models capture (see [17] for discussion). The shape of the saturation is different for neural (Fig. 2a-b) versus perceptual (Fig. 5b-c) data, in both experiment and model; for the latter, this arises from differences in stimuli (gratings versus bars, how the center and surround extents were determined).

The second class of saliency effects involves collinear facilitation. One example is the so called *border effect*, shown in figure 6a – one side of the border, whose individual bars are collinear, is more salient than the other (e.g. [1], but see also [40]). The middle and right plots in figure 6a depict the saliency map for the full model and a reduced model that uses a diagonal covariance matrix. Notice that the reduced model also shows an enhancement of the collinear side of the border *vs* the parallel, due to the partial overlap of the linear receptive fields; but, as explained in Sec. 2.3, the higher covariance between collinear filters in the full model, strengthens the effect. To quantify the difference, we report also the ratio between the salience values on the collinear and parallel sides of the border, after subtracting the saliency value of the homogeneous regions: the lower value for the reduced model (1.28; versus 1.74 for the full model) shows that the full model enhances the collinear relative to the parallel side. The ratio for the full model increases if we rescale the off–diagonal terms of the covariance matrix relative to the diagonal (2.1 for a scaling factor of 1.5; 2.73 for a factor of 2). Rescaling would come from more reasonably dense spatial sampling. Fig. 6b provides another, stronger example of the collinear facilitation.

## 5 Discussion

We have extended a standard GSM generative model of scene statistics to encompass contextual effects. We modeled the covariance between the Gaussian components associated with neighboring locations, and suggested a simple, approximate, process for choosing whether or not to pool such locations under the same mixer. Using parameters learned from natural scenes, we showed that this model provides a promising account of neurophysiological data on area summation and center-surround orientation contrast, and perceptual data on the saliency of image elements. This form of model has previously been applied to the tilt illusion [34], but had just *assumed* the assignments of figure 3c, in order to account for the indirect tilt illusion. Here, this emerged from first principles. This model therefore unifies a wealth of data and ideas about contextual visual processing. To our

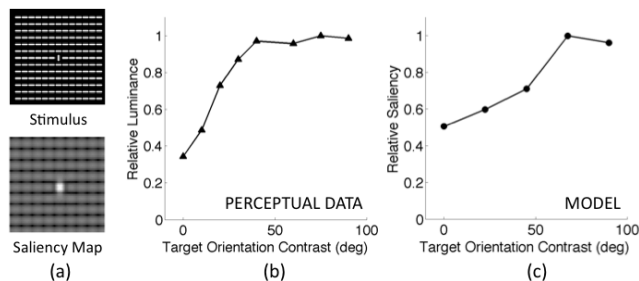

Figure 5: (a) An example of the stimulus and saliency map computed by the model. (b) Perceptual data reproduced after [8], and (c) model output, of the saliency of the central bar as a function of the orientation contrast between center and surround.

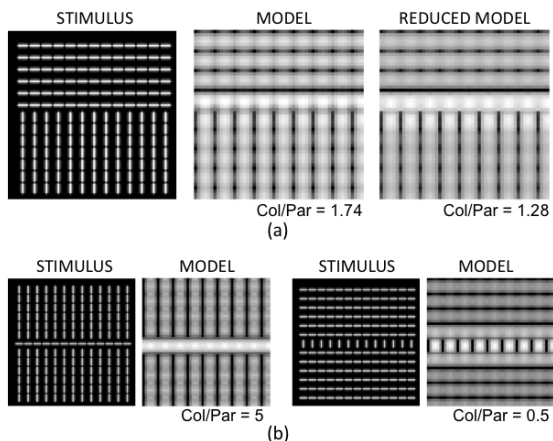

Figure 6: (a) Border effect: the collinear side of the border is more salient than the parallel one; the center plot is the saliency map for the full model, right plot is for a reduced model with diagonal covariance matrix. (b) Another example of collinear facilitation: the center row of bars is more salient, relative to the background, when the bars are collinear (left) rather than when they are parallel (right). In both (a) and (b), $Col/Par$ is the ratio between the salience values on the collinear and parallel sides of the border, after subtracting the saliency value of the homogeneous regions.

knowledge, there have only been few previous attempts of this sort; one notable example is the extensive salience work of [17]; here we go further in terms of simulating neural non–linearities, and making connections with the contour integration and illusion literature: phenomena that have previously been addressed only individually, if at all.

Our model is closely related to a number of suggestions in the literature. Previous bottom-up models of divisive normalization, which were the original inspiration for the application by [22] of the GSM, can account for some neural non–linearities by learning divisive weights instead of assignments (e.g., [33]). However they do not incorporate linear correlations, and they fix the divisive weights a priori rather than on a image–by–image basis such as in our model. Non-parametric statistical alternatives to divisive normalization, e.g. non-linear ICA [41], have also been proposed, but have been applied only to the orientation masking nonlinearity, therefore not addressing spatial context.

There are also various top-down models based on related principles. Compared with previous GSM modelling [30], we have built a more computationally straightforward, and neurobiologically credible, approximate assignment mechanism. Other recent generative statistical models that capture the statistical dependencies of the filters in slightly different ways (notably [31, 35]), might also be able to encompass the data we have presented here. However, [35] has been applied only to the image processing domain, and the model of [31] has not been tied to the perceptual phenomena we have considered, nor to contrast data. There are also quantitative differences between the models, including issues of soft versus hard assignment (see discussion in [30]); the assumption about the link to data (here we adopted the mean of the Gaussian component of the GSM which incorporates

an explicit gain control, in contrast to the approach in [31]); and the richness of assignment versus approximation in the various models (here we have purposely taken an approximate version of a full assignment model).

There are also many models devoted to saliency. We showed that our assignment process, and the normalization that results, is a good match for (and thus a normative justification of) at least some of the results that [1, 15] captured in a dynamical realization of the V1 saliency hypothesis. However, our model achieves suppression in regions of statistical homogeneity divisively rather than subtractively. The covariance between the Gaussian components captures some aspects of the long range excitatory effects in that model, which permit contour integration. However, some of the collinear facilitation arises just from receptive field overlap; and the structure of the covariance in natural scenes seems rather impoverished compared with that implied by the association field [42], and merits further examination with higher order statistics (see also [10, 26]). Note also that dynamical models have not previously been applied to the same range of data (such as the tilt illusion).

Open theoretical issues include quantifying carefully the effect of the rather coarse assignment approximation, as well as the differences between the learned model and the idealized population model of the tilt illusion [34]. Other important issues include characterizing the nature and effect of uncertainty in the distributions of $\mathbf{g}$ and $v$ rather than just the mean. This is critical to characterize psychophysical results on contrast detection in the face of noise and also orientation acuity, and also raises the issues aired by [31] as to how neural responses convey uncertainties. Open experimental issues include a range of other contextual effects as to salience, contour integration, and even perceptual crowding. Contextual effects are equally present at multiple levels of neural processing. An important future generalization would be to higher neural areas, and to mid and high level vision (which themselves exhibit gain-control related phenomena, see e.g. [43]). More generally, context is pervasive in time as well as space. The parallels are underexplored, and so pressing.

## Acknowledgements

This work was funded by the Alfred P. Sloan Foundation (OS); and The Gatsby Charitable Foundation, the BBSRC, the EPSRC and the Wellcome Trust (PD). We are very grateful to Adam Kohn, Joshua Solomon, Adam Sanborn, and Li Zhaoping for discussion.

## Footnotes

[1] To ensure that the mixer follows the same distribution under $\xi_1$ and $\xi_2$, after training with natural images we rescale $v_c$ — and therefore $\mathbf{g}_c$ — so that their values span the same range in both configurations; since the assignment is made at a higher level in the hierarchy, such a rescaling is equivalent to downstream normalization processes that make the estimates of $\mathbf{g}_c$ comparable under $\xi_1$ and $\xi_2$.

## References

[1] Z. Li. A saliency map in primary visual cortex. *Trends Cogn Sci*, 6(1):9–16, 2002.

[2] P. Series, J. Lorenceau, and Y. Frégnac. The "silent" surround of v1 receptive fields: theory and experiments. *J Physiol Paris*, 97(4-6):453–474, 2003.

[3] H. E. Jones, K. L. Grieve, W. Wang, and A. M. Sillito. Surround suppression in primate v1. *J Neurophysiol*, 86(4):2011–2028, 2001.

[4] J. R. Cavanaugh, W. Bair, and J. A. Movshon. Selectivity and spatial distribution of signals from the receptive field surround in macaque v1 neurons. *J Neurophysiol*, 88(5):2547–2556, 2002.

[5] J. J. Gibson and M. Radner. Adaptation, after-effect, and contrast in the perception of tilted lines. *Journal of Experimental Psychology*, 20:553–569, 1937.

[6] C. W. Clifford, P. Wenderoth, and B. Spehar. A functional angle on some after-effects in cortical vision. *Proc R Soc Lond B Biol Sci*, 1454:1705–1710, 2000.

[7] J. A. Solomon and M. J. Morgan. Stochastic re-calibration: contextual effects on perceived tilt. *Proc Biol Sci*, 273(1601):2681–2686, 2006.

[8] H.C. Nothdurft. The conspicuousness of orientation and motion contrast. *Spatial Vision*, 7(4):341–363, 1993.

[9] D. J. Field, A. Hayes, and R. F. Hess. Contour integration by the human visual system: evidence for a local "association field". *Vision Res*, 33(2):173–193, 1993.

[10] W. S. Geisler, J. S. Perry, B. J. Super, and D. P. Gallogly. Edge co-occurrence in natural images predicts contour grouping performance. *Vision Res*, 41(6):711–724, 2001.

[11] L. Schwabe, K. Obermayer, A. Angelucci, and P. C. Bressloff. The role of feedback in shaping the extraclassical receptive field of cortical neurons: a recurrent network model. *J Neurosci*, 26(36):9117–9129, 2006.

[12] J. Wielaard and P. Sajda. Extraclassical receptive field phenomena and short-range connectivity in v1. *Cereb Cortex*, 16(11):1531–1545, 2006.

[13] T. J. Sullivan and V. R. de Sa. A model of surround suppression through cortical feedback. *Neural Netw*, 19(5):564–572, 2006.

[14] T.N. Mundhenk and L. Itti. Computational modeling and exploration of contour integration for visual saliency. *Biological Cybernetics*, 93(3):188–212, 2005.

[15] Z. Li. Visual segmentation by contextual influences via intracortical interactions in primary visual cortex. *Network: Computation in Neural Systems*, 10(2):187–212, 1999.

[16] L. Itti and C. Koch. A saliency-based search mechanism for overt and covert shifts of visual attention. *Vision Research*, 40(10-12):1489–1506, 2000.

[17] D. Gao, V. Mahadevan, and N. Vasconselos. On the plausibility of the discriminant center-surround hypothesis for visual saliency. *Journal of Vision*, 8(7)(13):1–18, 2008.

[18] L. Zhang, M.H. Tong, T. Marks, H. Shan, and G.W. Cottrell. Sun: A bayesian framework for saliency using natural statistics. *Journal of Vision*, 8(7)(32):1–20, 2008.

[19] N.D.B. Bruce and J.K. Tsotsos. Saliency, attention, and visual search: An information theoretic approach. *Journal of Vision*, 9(3)(5):1–24, 2009.

[20] A. Hyvärinen, J. Hurri, and P.O. Hoyer. *Natural Image Statistics*. Springer, 2009.

[21] D. Andrews and C. Mallows. Scale mixtures of normal distributions. *J. Royal Stat. Soc.*, 36:99–102, 1974.

[22] M. J. Wainwright, E. P. Simoncelli, and A. S. Willsky. Random cascades on wavelet trees and their use in modeling and analyzing natural imagery. *Applied and Computational Harmonic Analysis*, 11(1):89–123, 2001.

[23] J. Portilla, V. Strela, M. Wainwright, and E. P. Simoncelli. Image denoising using a scale mixture of Gaussians in the wavelet domain. *IEEE Trans Image Processing*, 12(11):1338–1351, 2003.

[24] C. Zetzsche, B. Wegmann, and E. Barth. Nonlinear aspects of primary vision: Entropy reduction beyond decorrelation. In *Int'l Symposium, Society for Information Display*, volume XXIV, pages 933–936, 1993.

[25] E. P. Simoncelli. Statistical models for images: Compression, restoration and synthesis. In *Proc 31st Asilomar Conf on Signals, Systems and Computers*, pages 673–678, Pacific Grove, CA, 1997. IEEE Computer Society.

[26] P. Hoyer and A. Hyvärinen. A multi-layer sparse coding network learns contour coding from natural images. *Vision Research*, 42(12):1593–1605, 2002.

[27] A Hyvärinen, J. Hurri, and J. Vayrynen. Bubbles: a unifying framework for low-level statistical properties of natural image sequences. *Journal of the Optical Society of America A*, 20:1237–1252, 2003.

[28] Y. Karklin and M. S. Lewicki. A hierarchical Bayesian model for learning nonlinear statistical regularities in nonstationary natural signals. *Neural Computation*, 17:397–423, 2005.

[29] S. Osindero, M. Welling, and G. E. Hinton. Topographic product models applied to natural scene statistics. *Neural Computation*, 18(2):381–414, 2006.

[30] O. Schwartz, T. J. Sejnowski, and P. Dayan. Soft mixer assignment in a hierarchical generative model of natural scene statistics. *Neural Comput*, 18(11):2680–2718, 2006.

[31] Y. Karklin and M.S. Lewicki. Emergence of complex cell properties by learning to generalize in natural scenes. *Nature*, 457(7225):83–86, 2009.

[32] D. J. Heeger. Normalization of cell responses in cat striate cortex. *Visual Neuroscience*, 9:181–198, 1992.

[33] O. Schwartz and E. P. Simoncelli. Natural signal statistics and sensory gain control. *Nature Neuroscience*, 4(8):819–825, 2001.

[34] O. Schwartz, T.J. Sejnowski, and P. Dayan. Perceptual organization in the tilt illusion. *Journal of Vision*, 9(4)(19):1–20, 2009.

[35] J.A. Guerrero-Colon, E.P. Simoncelli, and J. Portilla. Image denoising using mixtures of gaussian scale mixtures. In *Proc 15th IEEE Int'l Conf on Image Proc*, pages 565–568, 2008.

[36] E. P. Simoncelli, W. T. Freeman, E. H. Adelson, and D. J. Heeger. Shiftable multi-scale transforms. *IEEE Trans Information Theory*, 38(2):587–607, 1992.

[37] C. K. I. Williams and N. J. Adams. Dynamic trees. In M. S. Kearns, S. A. Solla, and D. A. Cohn, editors, *Adv. Neural Information Processing Systems*, volume 11, pages 634–640, Cambridge, MA, 1999. MIT Press.

[38] X.L. Meng and D.B. Rubin. Maximum likelihood estimation via the ecm algorithm: A general framework. *Biometrika*, 80(2):267–278, 1993.

[39] E. Goddard, C.W.G. Clifford, and S.G. Solomon. Centre-surround effects on perceived orientation in complex images. *Vision Research*, 48:1374–1382, 2008.

[40] A.V. Popple and Z. Li. Testing a v1 model: perceptual biases and saliency effects. *Journal of Vision*, 1,3:148, 2001.

[41] J. Malo and J. Gutiérrez. V1 non–linear properties emerge from local–to–global non–linear ica. *Network: Comp. Neur. Syst.*, 17(1):85–102, 2006.

[42] D. J. Field. Relations between the statistics of natural images and the response properties of cortical cells. *J. Opt. Soc. Am. A*, 4(12):2379–2394, 1987.

[43] Q. Li and Z. Wang. General–purpose reduced–reference image quality assessment based on perceptually and statistically motivated image representation. In *Proc 15th IEEE Int'l Conf on Image Proc*, pages 1192–1195, 2008.

